# Transfer Learning by Borrowing Examples for Multiclass Object Detection

**Joseph J. Lim**
CSAIL, MIT
lim@csail.mit.edu

**Ruslan Salakhutdinov**
Department of Statistics, University of Toronto
rsalakhu@utstat.toronto.edu

**Antonio Torralba**
CSAIL, MIT
torralba@csail.mit.edu

## Abstract

Despite the recent trend of increasingly large datasets for object detection, there still exist many classes with few training examples. To overcome this lack of training data for certain classes, we propose a novel way of augmenting the training data for each class by borrowing and transforming examples from other classes. Our model learns which training instances from other classes to borrow and how to transform the borrowed examples so that they become more similar to instances from the target class. Our experimental results demonstrate that our new object detector, with borrowed and transformed examples, improves upon the current state-of-the-art detector on the challenging SUN09 object detection dataset.

## 1 Introduction

Consider building a *sofa* detector using a database of annotated images containing sofas and many other classes, as shown in Figure 1. One possibility would be to train the sofa detector using only the sofa instances. However, this would result in somewhat poor performance due to the limited size of the training set. An alternative is to build priors about the appearance of object categories and share information among object models of different classes. In most previous work, transfer of information between models takes place by imposing some regularization across model parameters. This is the standard approach both in the discriminative setting [1, 2, 3, 4, 5, 6, 7, 8] and in generative object models [9, 10, 11, 12, 13, 14].

In this paper, we propose a different approach to transfer information across object categories. Instead of building object models in which we enforce regularization across the model parameters, we propose to directly share training examples from similar categories. In the example from Figure 1, we can try to use training examples from other classes that are similar enough, for instance *armchairs*. We could just add all the armchair examples to the sofa training set. However, not all instances of armchairs will look close enough to sofa examples to train an effective detector. Therefore, we propose a mechanism to select, among all training examples from other classes, which ones are closer to the *sofa* class. We can increase the number of instances that we can *borrow* by applying various transformations (e.g., stretching armchair instances horizontally to look closer to sofas). The transformations will also depend on the viewpoint. For instance, a frontal view of an armchair looks like a compressed sofa, whereas the side view of an armchair and a sofa often look indistinguishable. Our approach differs from generating new examples by perturbing examples (e.g., adding mirrored or rotated versions) from its own class [15]. Rather, these techniques can be combined with ours.

Our approach looks for the set of classes to borrow from, which samples to borrow, and what the best transformation for each example is. Our work has similarities with three pieces of work on transfer

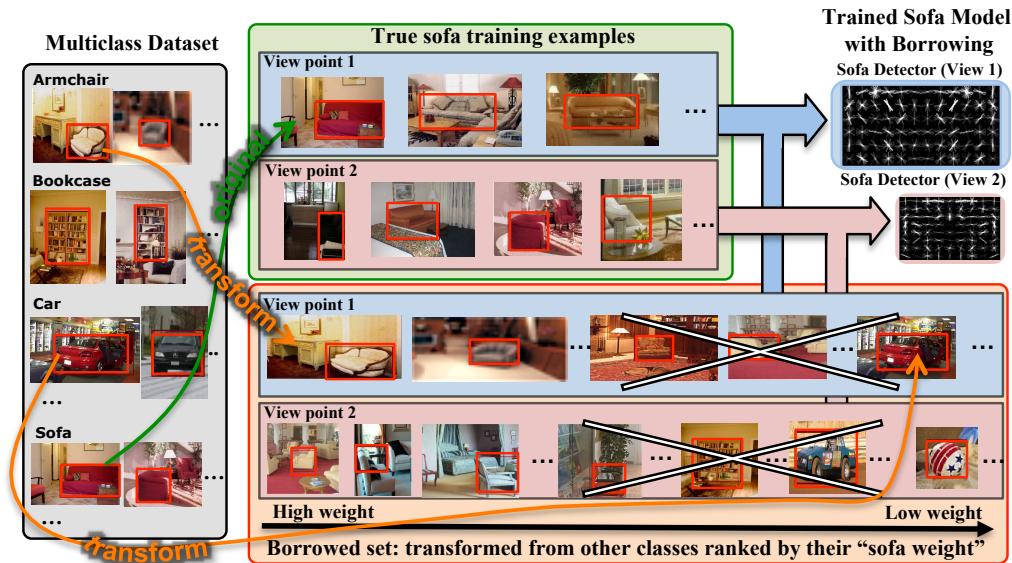

Figure 1: An illustration of training a sofa detector by borrowing examples from other related classes. Our model can find (1) good examples to borrow, by *learning* a weight for each example, and (2) the best transformation for each training example in order to increase the borrowing flexibility. Transformed examples in blue (or red) box are more similar to the sofa's frontal (or side) view. Transformed examples, which are selected according to their learned weights, are trained for sofa together with the original sofa examples. (X on images indicates that they have low weights to be borrowed)

learning for object recognition. Miller et al. [9] propose a generative model for digits that shares transformations across classes. The generative model decomposes each model into an appearance model and a distribution over transformations that can be applied to the visual appearance to generate new samples. The set of transformations is shared across classes. In their work, the transfer of information is achieved by sharing parameters across the generative models and not by reusing training examples. The work by Fergus et al. [16] achieves transfer across classes by learning a regression from features to labels. Training examples from classes similar to the target class are assigned labels between +1 and −1. This is similar to borrowing training examples but relaxing the confidence of the classification score for the borrowed examples. Wang et al. [17] assign rankings to similar examples, by enforcing the highest and lowest rankings for the original positive and negative examples, respectively, and requiring borrowed examples be somewhere in between. Both of these works rely on a pre-defined similarity metric (e.g. WordNet or aspect based similarity) for deciding which classes to share with. Our method, on the other hand, learns which classes to borrow from as well as which examples to borrow within those classes as part of the model learning process.

Borrowing training examples becomes effective when many categories are available. When there are few and distinct object classes, as in the PASCAL dataset [18], the improvement may be limited. However, a number of other efforts are under way for building large annotated image databases with many categories [19, 20, 21]. As the number of classes grows, the number of sets of classes with similar visual appearances (e.g., the set of *truck, car, van, suv*, or *chair, armchair, swivel chair, sofa*) will increase, and the effectiveness of our approach will grow as well. In our experiments, we show that borrowing training examples from other classes results in improved performance upon the current state of the art detectors trained on a single class. In addition, we also show that our technique can be used in a different but related task. In some cases, we are interested in merging multiple datasets in order to improve the performance on a particular test set. We show that learning examples to merge results in better performance than simply combining the two datasets.

## 2   Learning to Borrow Examples

Consider the challenging problem of detecting and localizing objects from a wide variety of categories such as cars, chairs, and trees. Many current state-of-the-art object detection (and object recognition) systems use rather elaborate models, based on separate appearance and shape components, that can cope with changes in viewpoint, illumination, shape and other visual properties. However, many of these systems [22, 23] detect objects by testing sub-windows and scoring corre-

sponding image patches $\mathbf{x}$ with a *linear function* of the form: $y = \boldsymbol{\beta}^\top \Phi(\mathbf{x})$, where $\Phi(\mathbf{x})$ represents a vector of different image features, and $\boldsymbol{\beta}$ represents a vector of model parameters.

In this work, we focus on training detection systems for multiple object classes. Our goal is to develop a novel framework that enables borrowing examples from related classes for a generic object detector, making minimal assumptions about the type of classifier, or image features used.

## 2.1 Loss Function for Borrowing Examples

Consider a classification problem where we observe a dataset $\mathcal{D} = \{\mathbf{x}_i, y_i\}_{i=1}^n$ of $n$ labeled training examples. Each example belongs to one of $C$ classes (e.g. 100 object classes), and each class $c \in \mathcal{C} = \{1, ..., C\}$ contains a set of $n_c$ labeled examples. We let $\mathbf{x}_i \in \mathrm{R}^D$ denote the input feature vector of length $D$ for the training case $i$, and $y_i$ be its corresponding class label. Suppose that we are also given a separate background class, containing $b$ examples. We further assume a binary representation for class labels[1], i.e. $y_i \in \mathcal{C} \cup \{-1\}$, indicating whether a training example $i$ belongs to one of the given $C$ classes, or the "negative" background class[2].

For a standard binary classification problem, a commonly used approach is to minimize:

$$\min_{\boldsymbol{\beta}^c} \left( \sum_{i=1}^{n_c+b} \texttt{Loss}\big(\boldsymbol{\beta}^c \cdot \mathbf{x}_i, \text{sign}(y_i)\big) + \lambda R(\boldsymbol{\beta}^c) \right), \tag{1}$$

where $i$ ranges over the positive and negative examples of the target class $c$; $\boldsymbol{\beta}^c \in \mathrm{R}^D$ is the vector of unknown parameters, or regression coefficients, for class $c$; $\texttt{Loss}(\cdot)$ is the associated loss function; and $R(\cdot)$ is a regularization function for $\boldsymbol{\beta}$.

Now, consider learning which other training examples from the entire dataset $\mathcal{D}$ our target class $c$ could borrow. The key idea is to learn a vector of weights $\mathbf{w}^c$ of length $n + b$, such that each $w_i^c$ would represent a soft indicator of how much class $c$ borrows from the training example $x_i$. Soft indicator variables $w_i^c$ will range between 0 and 1, with 0 indicating borrowing none and 1 indicating borrowing the entire example as an additional training instance of class $c$. All true positive examples belonging to class $c$, with $y_i = c$, and all true negative examples belonging to the background class, with $y_i = -1$, will have $w_i^c = 1$, as they will be used fully. Remaining training examples will have $w_i^c$ between 0 and 1. Our proposed regularization model takes the following form:

$$\sum_{c \in \mathcal{C}} \min_{\boldsymbol{\beta}^c} \min_{\mathbf{w}^{*,c}} \left( \sum_{i=1}^{n+b} (1 - w_i^{*,c}) \texttt{Loss}\big(\beta^c \cdot x_i, \text{sign}(y_i)\big) + \lambda R(\boldsymbol{\beta}^c) + \Omega_{\lambda_1, \lambda_2}(\mathbf{w}^{*,c}) \right), \tag{2}$$

subject to $w_i^c = 1$ for $y_i = -1$ *or* $c$, and $0 \le w_i^c \le 1$ for all other $i$, where we defined[3] $\mathbf{w}^* = 1 - \mathbf{w}$, and where $i$ ranges over *all training examples* in the dataset. We further define $\Omega(\mathbf{w}^*)$ as:

$$\Omega_{\lambda_1, \lambda_2}(\mathbf{w}^*) = \lambda_1 \sum_{l \in \mathcal{C}} \sqrt{n_l} \|\mathbf{w}_{(l)}^*\|_2 + \lambda_2 \|\mathbf{w}^*\|_1, \tag{3}$$

where $\mathbf{w}_{(l)}^*$ represents a vector of weights for class $l$, with $\mathbf{w}_{(l)}^* = (w_{j_1}^*, w_{j_2}^*, \cdots, w_{j_{n_l}}^*)$ for $y_{j_m} = l$. Here, $\Omega(\cdot)$ regularizes $\mathbf{w}^{*,c}$ using a sparse group lasso criterion [24]. Its first term can be viewed as an intermediate between the $L_1$ and $L_2$-type penalty. A pleasing property of $L_1$-$L_2$ regularization is that it performs variable selection at the group level. The second term of $\Omega(\cdot)$ is an $L_1$-norm, which keeps the sparsity of weights at the individual level.

The overall objective of Eq (2) and its corresponding regularizer $\Omega(\cdot)$ have an intuitive interpretation. The regularization term encourages borrowing *all examples* as new training instances for the target class $c$. Indeed, setting corresponding regularization parameters $\lambda_1$ and $\lambda_2$ to high enough values (i.e. forcing $\mathbf{w}$ to be an all $\mathbf{1}$ vector) would amount to borrowing all examples, which would result in learning a "generic" object detector. On the other hand, setting $\lambda_1 = \lambda_2 = 0$ would recover the original standard objective of Eq (1), without borrowing any examples. Figure 2b displays learned $w_i$ for 6547 instances to be borrowed by the *truck* class. Observe that classes that have similar visual appearances to the target *truck* class (e.g. *van*, *bus*) have $w_i$ close to 1 and are grouped together (compare with Figure 2a, which only uses an $L_1$ norm).

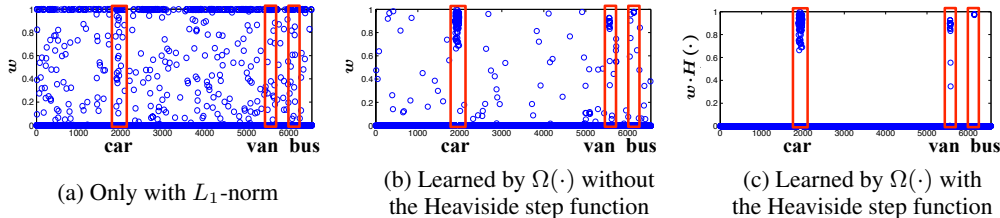

(a) Only with $L_1$-norm

(b) Learned by $\Omega(\cdot)$ without the Heaviside step function

(c) Learned by $\Omega(\cdot)$ with the Heaviside step function

Figure 2: **Learning to borrow for the target truck class:** Learned weights $\mathbf{w}^{\text{truck}}$ for 6547 instances using (a) $L_1$-norm; (b) $\Omega(\cdot)$ regularization; and (c) $\Omega(\cdot)$ with symmetric borrowing constraint.

We would also like to point out an analogy between our model and various other transfer learning models that regularize the $\boldsymbol{\beta}$ parameter space [25, 26]. The general form applied to our problem setting takes the following form:

$$\sum_{c \in \mathcal{C}} \min_{\boldsymbol{\beta}^c} \left( \sum_i \texttt{Loss}(\boldsymbol{\beta}^c \cdot \mathbf{x}_i, \text{sign}(y_i)) + \lambda R(\boldsymbol{\beta}^c) + \gamma \| \boldsymbol{\beta}^c - \frac{1}{C} \sum_{k=1}^{C} \boldsymbol{\beta}^k \|_2^2 \right). \tag{4}$$

The model in Eq (4) regularizes all $\boldsymbol{\beta}^c$ to be close to a single mode, $\frac{1}{C} \sum_k \boldsymbol{\beta}^k$. This can be further generalized so that $\boldsymbol{\beta}^c$ is regularized toward one of many modes, or "super-categories", as pursued in [27]. Contrary to previous work, our model from Eq (2) regularizes weights on all training examples, rather than parameters, across all categories. This allows us to *directly learn* both: which examples and what categories we should borrow from. We also note that model performance could potentially be improved by introducing additional regularization across model parameters.

## 2.2 Learning

Solving our final optimization problem, Eq (2), for $\mathbf{w}$ and $\boldsymbol{\beta}$ jointly is a non-convex problem. We therefore resort to an iterative algorithm based on the fact that solving for $\boldsymbol{\beta}$ given $\mathbf{w}$ and for $\mathbf{w}$ given $\boldsymbol{\beta}$ are convex problems. The algorithm will iterate between (1) solving for $\boldsymbol{\beta}$ given $\mathbf{w}$ based on [22], and (2) solving for $\mathbf{w}$ given $\boldsymbol{\beta}$ using the block coordinate descent algorithm [28] until convergence. We initialize the model by setting $w_i^c$ to 1 for $y_i = c$ and $y_i = -1$, and to 0 for all other training examples. Given this initialization, the first iteration is equivalent to solving $C$ separate binary classification problems of Eq (1), when there is no borrowing[4]

Even though most irrelevant examples have low borrowing indicator weights $w_i$, it is ideal to clean up these noisy examples. To this end, we introduce a symmetric borrowing constraint: if a *car* class does not borrow examples from *chair* class, then we would also like for the *chair* class not to borrow examples from the corresponding *car* class. To accomplish this, we multiply $w_i^c$ by $H(\bar{w}_c^{y_i} - \epsilon)$, where $H(\cdot)$ is the Heaviside step function. We note that $w_i^c$ refers to the weight of example $x_i$ to be borrowed by the target class $c$, whereas $\bar{w}_c^{y_i}$ refers to the average weight of examples that class $y_i$ borrows from the target class $c$. In other words, if the examples that class $y_i$ borrows from class $c$ have low weights on average (i.e. $\bar{w}_c^{y_i} < \epsilon$), then class $c$ will not borrow example $x_i$, as this indicates that classes $c$ and $y_i$ may not be similar enough. The resulting weights after introducing this symmetric relationship are shown in Figure 2c.

# 3 Borrowing Transformed Examples

So far, we have assumed that each training example is borrowed as is. Here, we describe how we apply transformations to the candidate examples during the training phase. This will allow us to borrow from a much richer set of categories such as *sofa-armchair*, *cushion-pillow*, and *car-van*. There are three different transformations we employ: translation, scaling, and affine transformation.

**Translation and scaling:** Translation and scaling are naturally inherited into existing detection systems during scoring. Scaling is resolved by scanning windows at multiple scales of the image, which typical sliding-window detectors already do. Translation is implemented by relaxing the location of the ground-truth bounding box $B_i$. Similar to Felzenszwalb et al. [22]'s approach of finding latent positive examples, we extract $\mathbf{x}_i$ from multiple boxes that have a significant overlap with $B_i$, and select a candidate example that has the smallest $\texttt{Loss}(\boldsymbol{\beta}^c \cdot \mathbf{x}_i, \text{sign}(y_i))$.

| Original Class | Without transformation | | With transformation | |
|---|---|---|---|---|
| | Borrowed Classes | AP improvement | Borrowed Classes | AP improvement |
| Truck | car, van | +7.14 | car, van | +9.49 |
| Shelves | bookcase | +0.17 | bookcase | +4.73 |
| Car | truck, van | +1.07 | truck, van, **bus** | +1.78 |
| Desk lamp | ∅ | N/A | **floor lamp** | +0.30 |
| Toilet | ∅ | N/A | **sink**, **cup** | -0.68 |

Table 1: **Learned borrowing relationships**: Most discovered relations are consistent with human subjective judgment. Classes that were borrowed only with transformations are shown in bold.

**Affine transformation:** We also change aspect ratios of borrowed examples so that they look more alike (as in *sofa-armchair* and *desk lamp-floor lamp*). Our method is to transform training examples to every canonical aspect ratio of the target class $c$, and find the best candidate for borrowing. The canonical aspect ratios can be determined by clustering aspect ratios of all ground-truth bounding boxes [22], or based on the viewpoints, provided we have labels for each viewpoint. Specifically, suppose that there is a candidate example $\mathbf{x}_i$ to be borrowed by the target class $c$ and there are $L$ canonical aspect ratios of $c$. We transform $\mathbf{x}_i$ into $\mathbf{x}_i^l$ by resizing one dimension so that $\{\mathbf{x}_i^l\}_{0 \leq l \leq L}$ contains all $L$ canonical aspect ratios of $c$ (and $\mathbf{x}_i^0 = \mathbf{x}_i$). In order to ensure that only one candidate is generated from $\mathbf{x}_i$, we select a single transformed example $\mathbf{x}_i^l$, for each $i$, that minimizes $\texttt{Loss}(\boldsymbol{\beta}^c \cdot \mathbf{x}_i^l, \text{sign}(y_i))$. Note that this final candidate can be selected during every training iteration, so that the best selection can change as the model is updated.

Figure 1 illustrates the kind of learning our model performs. To borrow examples for *sofa*, each example in the dataset is transformed into the frontal and side view aspect ratios of *sofa*. The transformed example that has the smallest $\texttt{Loss}(\cdot)$ is selected for borrowing. Each example is then assigned a borrowing weight using Eq (2). Finally, the new *sofa* detector is trained using borrowed examples together with the original *sofa* examples. We refer the detector trained without affine transformation as the **borrowed-set** detector, and the one trained with affine transformation as the **borrowed-transformed** detector.

## 4 Experimental Results

We present experimental results on two standard datasets: the SUN09 dataset [21] and the PASCAL VOC 2007 challenge [18]. The SUN09 dataset contains 4,082 training images and 9,518 testing images. We selected the top 100 object categories according to the number of training examples. These 100 object categories include a wide variety of classes such as *bed*, *car*, *stool*, *column*, and *flowers*, and their distribution is heavy tailed varying from 1356 to 8 instances. The PASCAL dataset contains 2,051 training images and 5,011 testing images, belonging to 20 different categories. For both datasets, we use the PASCAL VOC 2008 evaluation protocol [18]. During the testing phase, in order to enable a direct comparison between various detectors, we measure the detection score of class $c$ as the mean Average Precision (AP) score across all positive images that belong to class $c$ and randomly sub-sampled negative images, so that the ratio between positive and negative examples remains the same across all classes.

Our experiments are based on one of the state-of-art detectors [22]. Following [22], we use a hinge loss for $\texttt{Loss}(\cdot)$ and a squared $L_2$-norm for $R(\cdot)$ in Eq (2), where every detector contains two root components. There are four controllable parameters: $\lambda$, $\lambda_1$, $\lambda_2$, and $\epsilon$ (see Eq (2)). We used the same $\lambda$ as in [22]. $\lambda_1$ and $\lambda_2$ were picked based on the validation set, and $\epsilon$ was set to $0.6$. In order to improve computation time, we threshold each weight $w_i$ so that it will either be $0$ or $1$.

We perform two kinds of experiments: (1) borrowing examples from other classes within the same dataset, and (2) borrowing examples from the same class that come from a *different* dataset. Both experiments require identifying which examples are beneficial to borrow for the target class.

### 4.1 Borrowing from Other Classes

We first tested our model to identify a useful set of examples to borrow from *other classes* in order to improve the detection quality on the SUN09 dataset. A unique feature of the SUN09 dataset is that all images were downloaded from the internet without making any effort to create a uniform distribution over object classes. We argue that this represents a much more realistic setting, in which some classes contain a lot of training data and many other classes contain little data.

(a) Shelves for Bookcase

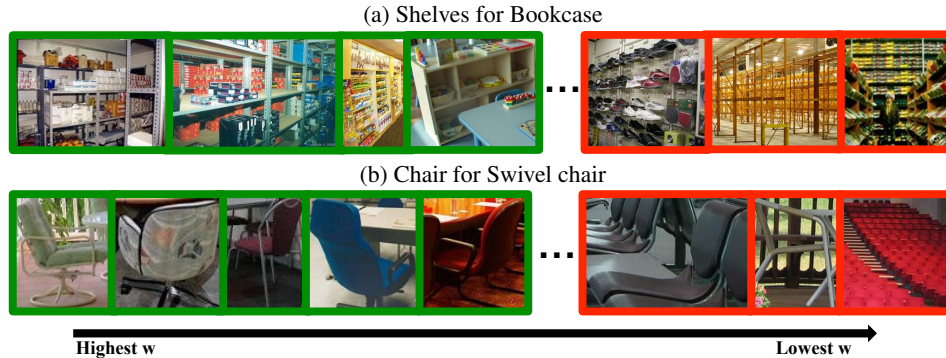

(b) Chair for Swivel chair

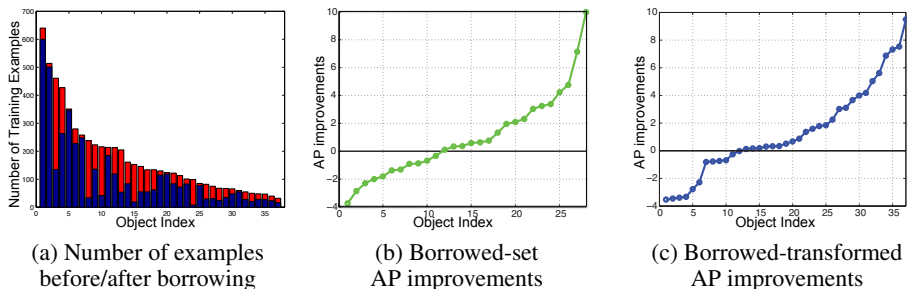

**Highest w**                                                                                                        **Lowest w**

Figure 3: **Borrowing Weights**: Examples are ranked by learned weights, $w$: (a) *shelves* examples to be borrowed by the *bookcase* class and (b) *chair* examples to be borrowed by the *swivel chair* class. Both show that examples with higher $w$ are more similar to the target class. (green: borrowed, red: not borrowed)

(a) Number of examples before/after borrowing

(b) Borrowed-set AP improvements

(c) Borrowed-transformed AP improvements

Figure 4: (a) Number of examples used for training per class before borrowing (blue) and after borrowing (red). Categories with fewer examples tend to borrow more examples. AP improvements (b) without and (c) with transformations, compared to the single detector trained only with the original examples. Note that our model learned to borrow from (b) 28 classes, and (c) 37 classes.

Among 100 classes, our model learned that there are 28 and 37 classes that can borrow from other classes without and with transformations, respectively. Table 1 shows some of the learned borrowing relationships along with their improvements. Most are consistent with human subjective judgment. Interestingly, our model excluded *bag*, *slot machine*, *flag*, and *fish*, among others, from borrowing. Many of those objects have quite distinctive visual appearances compared to other object categories.

Figure 3 shows borrowed examples along with their relative orders according to the borrowing indicator weights, $w_i$. Note that our model learns quite reliable weights: for example, *chair* examples in green box are similar to the target *swivel chair* class, whereas examples in red box are either occluded or very atypical.

Figure 4 further displays AP improvements of the borrowed-set and borrowed-transformed detectors, against standard single detectors. Observe that over 20 categories benefit in various degrees from borrowing related examples. Among borrowed-transformed detectors, the categories with the largest improvements are *truck* (9.49), *picture* (7.54), *bus* (7.32), *swivel chair* (6.88), and *bookcase* (5.62). We note that all of these objects borrow visual appearance from other related frequent objects, including *car*, *chair*, and *shelves*. Five objects with the largest decrease in AP include *plate* (-3.53), *fluorescent tube* (-3.45), *ball* (-3.21), *bed* (-2.69), and *microwave* (-2.52). Model performance often deteriorates when our model discovers relationships that are not ideal (e.g. *toilet* borrowing *cup* and *sink*; *plate* borrowing *mug*).

Table 2 further breaks down borrowing rates as a function of the number of training examples, where a borrowing rate is defined as the ratio of the total number of borrowed examples to the number of original training examples. Observe that borrowing rates are much higher when there are fewer training examples (see also Figure 4a). On average, the borrowed-set detectors borrow 75% of the total number of original training examples, whereas the borrowed-transformed detectors borrow about twice as many examples, 149%.

Table 3 shows AP improvements of our methods. Borrowed-set improve 1.00 and borrowed-transformed detectors improve 1.36. This is to be expected as introducing transformations allows us to borrow from a much richer set of object classes. We also compare to a baseline approach, which

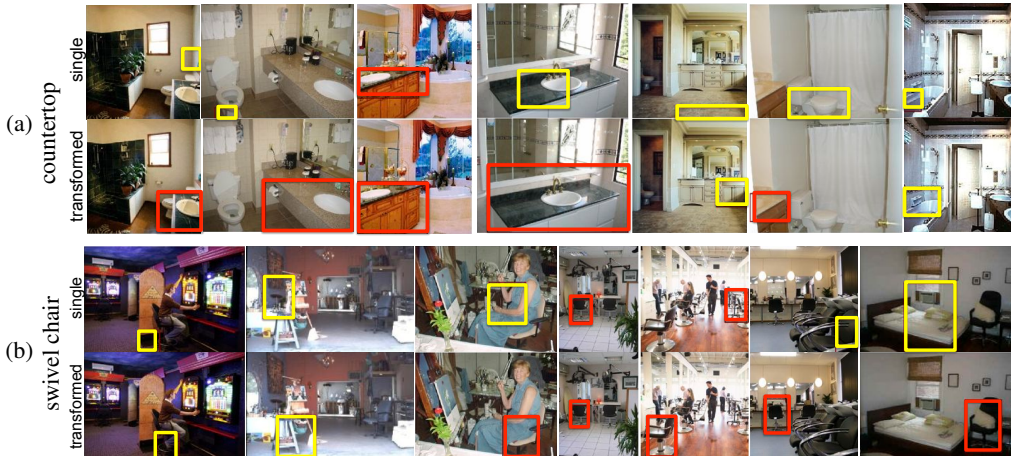

Figure 5: **Detection results** on random images containing the target class. Only the most confident detection is shown per image. For clearer visualization, we do not show images where both detectors have large overlap. Our detectors (2nd/4th row) show better localizations than single detectors (1st/3rd row). (red: correct detection, yellow: false detection)

| Number of Training Examples | 1-30 | 31-50 | 51-100 | 101-150 | > 150 | ALL |
|---|---|---|---|---|---|---|
| Borrowed-set | 1.69 | 0.48 | 0.43 | 0.48 | 0.13 | 0.75 |
| Borrowed-Transformed | 2.75 | 2.57 | 0.94 | 0.81 | 0.17 | 1.49 |

Table 2: **Borrowing rates** for the borrowed-set and borrowed-transformed models. Borrowing rate is defined as the ratio of the number of borrowed examples to the number of original examples.

| Methods | Borrowed-set | All examples from the same classes | Borrowed-Transformed |
|---|---|---|---|
| AP without borrowing | 14.99 | 16.59 | 16.59 |
| AP improvements | +1.00 | +0.30 | +1.36 |

Table 3: **AP improvements** of the borrowed-set and borrowed-transformed detectors. We also compared borrowed-transformed method against the baseline approach borrowing all examples, without any selection of examples, from the same classes our method borrows from. 2nd row shows the average AP score of the detectors without any borrowing in the classes used for borrowed-set or borrowed-transformed.

uses all examples in the borrowed classes of borrowed-transformed method. For example, if class A borrows some examples from class B and C using borrowed-transformed method, then the baseline approach uses all examples from class A, B, and C without any selection. Note that this baseline approach improves only 0.30 compared to 1.36 of our method.

Finally, Figure 5 displays detection results. Single and borrowed-transformed detections are visualized on test images, chosen at random, that contain the target class. In many cases, transformed detectors are better at localizing the target object, even when they fail to place a bounding box around the full object. We also note that borrowing similar examples tends to introduce some confusions between related object categories. However, we argue that this type of failure is much more tolerable compared to the single detector, which often has false detections of completely unrelated objects.

## 4.2   Borrowing from Other Datasets

Combining datasets is a non-trivial task as different datasets contain different biases. Consider training a car detector that is going to be evaluated on the PASCAL dataset. The best training set for such a detector would be the dataset provided by the PASCAL challenge, as both the training and test sets come from the same underlying distribution. In order to improve model performance, a simple mechanism would be to add additional training examples. For this, we could look for other datasets that contain annotated images of cars – for example, the SUN09 dataset. However, as the PASCAL and SUN09 datasets come with different biases, many of the training examples from SUN09 are not as effective for training when the detector is evaluated on the PASCAL dataset – a problem that was extensively studied by [29]. Here, we show that, instead of simply mixing the two datasets, our model can select a useful set of examples from the SUN09 for the PASCAL dataset, and vice-versa.

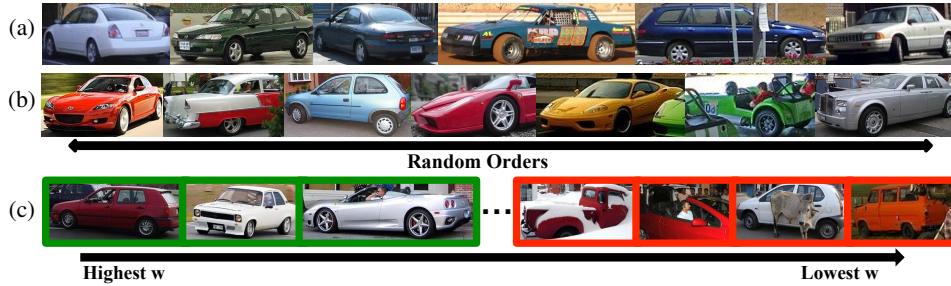

Random Orders

Highest w                                                                    Lowest w

Figure 6: **SUN09 borrowing PASCAL examples**: (a) Typical SUN09 car images, (b) Typical PASCAL car images, (c) PASCAL car images sorted by learned borrowing weights. (c) shows that examples are sorted from canonical view points (left) to atypical or occluded examples (right). (green: borrowed, red: not borrowed)

|  | SUN09 only | PASCAL only | SUN09 +PASCAL | SUN09 +borrow PASCAL |
|---|---|---|---|---|
| car | 43.31 | 39.47 | 43.64 | **45.88** |
| person | 45.46 | 28.78 | 46.46 | **46.90** |
| sofa | 12.96 | 11.97 | 12.86 | **15.25** |
| chair | 18.82 | 13.84 | 18.18 | **20.45** |
| mean | 30.14 | 23.51 | 30.29 | **32.12** |
| Diff. |  | -6.63 | +0.15 | **+1.98** |

(a) Testing on the SUN09 dataset

|  | PASCAL only | SUN09 only | PASCAL +SUN09 | PASCAL +borrow SUN09 |
|---|---|---|---|---|
| car | 49.58 | 40.81 | 49.91 | **51.00** |
| person | 23.58 | 22.31 | 26.05 | **27.05** |
| sofa | 19.91 | 13.99 | 20.01 | **22.17** |
| chair | 14.23 | 14.20 | **19.06** | 18.55 |
| mean | 26.83 | 22.83 | 28.76 | **29.69** |
| Diff. |  | -4.00 | +1.93 | **+2.86** |

(b) Testing on the PASCAL 2007 dataset

Table 4: **Borrowing from other datasets**: AP scores of various detectors: "SUN09 only" and "PASCAL only" are trained using the SUN09 dataset [21] and the PASCAL dataset [18] without borrowing any examples. "SUN09+PASCAL" is trained using positive examples from both SUN09 and PASCAL. and negative examples from the target dataset. "PASCAL+borrow SUN09" and "SUN09+borrow PASCAL" borrow selected examples from another dataset for each target dataset using our method. The last Diff row shows AP improvements over the "standard" *state-of-art* detector trained on the target dataset (*column 1*).

Figure 6 shows the kind of borrowing our model performs. Figure 6a,b display typical car images from the SUN09 and PASCAL datasets. Compared to SUN09, PASCAL images display a much wider variety of car types, with different viewpoints and occlusions. Figure 6c further shows the ranking of PASCAL examples by $w_i^{\text{SUN09 car}}$ for $i \in \mathcal{D}_{\text{PASCAL}}$. Observe that images with high $w$ match the canonical representations of SUN09 images much better compared to images with low $w$.

Table 4 shows performances of four detectors. Observe that detectors trained on the target dataset (*column 1*) outperform ones trained using another dataset (*column 2*). This shows that there exists a significant difference between the two datasets, which agrees with previous work [29]. Next, we tested detectors by simply combining positive examples from both datasets and using negative examples from the target dataset (*column 3*). On the SUN09 test set, the improvement was not significant, and on the PASCAL test set, we observed slight improvements. Detectors trained by our model (*column 4*) substantially outperformed single detectors as well as ones that were trained mixing the two datasets. The detectors (*columns 1* and *2*) were trained using the *state-of-art* algorithm [22].

## 5 Conclusion

In this paper we presented an effective method for transfer learning across object categories. The proposed approach consists of searching similar object categories using sparse grouped Lasso framework, and borrowing examples that have similar visual appearances to the target class. We further demonstrated that our method, both with and without transformation, is able to find useful object instances to borrow, resulting in improved accuracy for multi-class object detection compared to the state-of-the-art detector trained only with examples available for each class.

**Acknowledgments**: This work is funded by ONR MURI N000141010933, CAREER Award No. 07471 20, NSERC, and NSF Graduate Research Fellowship.

## Footnotes

[1]This is a standard "1 *vs.* all" classification setting.

[2]When learning a model for class $c$, all other classes can be considered as "negative" examples. In this work, for clarity of presentation, we will simply assume that we are given a separate background class.

[3]For clarity of presentation, throughout the rest of the paper, we will use the following identity $\mathbf{w}^* = 1 - \mathbf{w}$.

[4]In this paper, we iterate only once, as it was sufficient to borrow similar examples (see Figure 2).

## References

[1] Y. LeCun, L. Bottou, Y. Bengio, and P. Haffner. Gradient-based learning applied to document recognition. *Proceedings of the IEEE*, 86(11):2278–2324, November 1998.

[2] S. Krempp, D. Geman, and Y. Amit. Sequential learning of reusable parts for object detection. Technical report, CS Johns Hopkins, 2002.

[3] A. Torralba, K. P. Murphy, and W. T. Freeman. Sharing features: efficient boosting procedures for multi-class object detection. In *CVPR*, 2004.

[4] E. Bart and S. Ullman. Cross-generalization: learning novel classes from a single example by feature replacement. In *CVPR*, 2005.

[5] A. Opelt, A. Pinz, and A. Zisserman. Incremental learning of object detectors using a visual shape alphabet. In *CVPR*, 2006.

[6] K. Levi, M. Fink, and Y. Weiss. Learning from a small number of training examples by exploiting object categories. In *Workshop of Learning in Computer Vision*, 2004.

[7] A. Quattoni, M. Collins, and T.J. Darrell. Transfer learning for image classification with sparse prototype representations. In *CVPR*, 2008.

[8] C.H. Lampert, H. Nickisch, and S. Harmeling. Learning to detect unseen object classes by between-class attribute transfer. In *CVPR*, 2009.

[9] E. Miller, N. Matsakis, and P. Viola. Learning from one example through shared densities on transforms. In *CVPR*, 2000.

[10] L. Fei-Fei, R. Fergus, and P. Perona. A bayesian approach to unsupervised one-shot learning of object categories. In *ICCV*, 2003.

[11] L. Fei-Fei, R. Fergus, and P. Perona. Learning generative visual models from few training examples: an incremental bayesian approach tested on 101 object categories. In *IEEE. Workshop on GMBV*, 2004.

[12] E. Sudderth, A. Torralba, W. T. Freeman, and W. Willsky. Learning hierarchical models of scenes, objects, and parts. In *ICCV*, 2005.

[13] J. Sivic, B.C. Russell, A. Zisserman, W.T. Freeman, and A.A. Efros. Unsupervised discovery of visual object class hierarchies. In *CVPR*, 2008.

[14] E. Bart, I. Porteous, P. Perona, and M. Welling. Unsupervised learning of visual taxonomies. In *CVPR*, 2008.

[15] D.M. Gavrila and J. Giebel. Virtual sample generation for template-based shape matching. In *CVPR*, 2001.

[16] R. Fergus, H. Bernal, Y. Weiss, and A. Torralba. Semantic label sharing for learning with many categories. In *ECCV*, 2010.

[17] Gang Wang, David Forsyth, and Derek Hoiem. Comparative object similarity for improved recognition with few or no examples. In *CVPR*, 2010.

[18] M. Everingham, L. Van Gool, C. K. I. Williams, J. Winn, and A. Zisserman. The pascal visual object classes (voc) challenge. *International Journal of Computer Vision*, 88(2):303–338, June 2010.

[19] B. C. Russell, A. Torralba, K. P. Murphy, and W. T. Freeman. LabelMe: a database and web-based tool for image annotation. *IJCV*, 77(1-3):157–173, 2008.

[20] J. Deng, W. Dong, R. Socher, L.-J. Li, K. Li, and L. Fei-Fei. Imagenet: A large-scale hierarchical image database. In *CVPR*, 2009.

[21] Jianxiong Xiao, James Hays, Krista A. Ehinger, Aude Oliva, and Antonio Torralba. Sun database: Large-scale scene recognition from abbey to zoo. In *CVPR*, 2010.

[22] P.F. Felzenszwalb, R.B. Girshick, D. McAllester, and D. Ramanan. Object detection with discriminatively trained part-based models. *TPAMI*, 32(9):1627 –1645, 2010.

[23] N. Dalal and B. Triggs. Histograms of oriented gradients for human detection. In *CVPR*, 2005.

[24] Ming Yuan and Yi Lin. Model selection and estimation in regression with grouped variables. *Journal of the Royal Statistical Society, Series B*, 68:49–67, 2006.

[25] Theodoros Evgeniou and Massimiliano Pontil. Regularized multi–task learning. In *ACM SIGKDD*, 2004.

[26] T. Tommasi, F. Orabona, and B. Caputo. Safety in numbers: Learning categories from few examples with multi model knowledge transfer. In *CVPR*, 2011.

[27] R. Salakhutdinov, A. Torralba, and J. Tenenbaum. Learning to share visual appearance for multiclass object detection. In *CVPR*, 2011.

[28] Jerome Friedman, Trevor Hastie, , and Robert Tibshirani. A note on the group lasso and a sparse group lasso. *Technical report, Department of Statistics, Stanford University*, 2010.

[29] A. Torralba and A. Efros. Unbiased look at dataset bias. In *CVPR*, 2011.

